# Exploration in Model-based Reinforcement Learning by Empirically Estimating Learning Progress

**Manuel Lopes**
INRIA
Bordeaux, France

**Tobias Lang**
FU Berlin
Germany

**Marc Toussaint**
FU Berlin
Germany

**Pierre-Yves Oudeyer**
INRIA
Bordeaux, France

## Abstract

Formal exploration approaches in model-based reinforcement learning estimate the accuracy of the currently learned model without consideration of the empirical prediction error. For example, PAC-MDP approaches such as R-MAX base their model certainty on the amount of collected data, while Bayesian approaches assume a prior over the transition dynamics. We propose extensions to such approaches which drive exploration solely based on empirical estimates of the learner's accuracy and learning progress. We provide a "sanity check" theoretical analysis, discussing the behavior of our extensions in the standard stationary finite state-action case. We then provide experimental studies demonstrating the robustness of these exploration measures in cases of non-stationary environments or where original approaches are misled by wrong domain assumptions.

## 1 Introduction

Reinforcement learning (RL) agents need to solve the exploitation-exploration tradeoff. They have to exploit their current model of the environment. At the same time they need to explore the environment sufficiently to learn more about its reward-relevant structure. Established model-based approaches like $E^3$ [8] and R-MAX [4] take into account how often a state-action pair has been visited. With an efficient model learner, the estimated transition model can be guaranteed to be approximately correct after a sufficient, and efficient, number of visitations in a stationary domain. An alternative approach to exploration is Bayesian RL [11]. Bayesian RL exploits prior knowledge about the transition dynamics to reason explicitly about the uncertainty of the estimated model. Interestingly, these existing approaches estimate the accuracy of the currently learned model based only on visitation counts. They do not consider the actual *empirical* prediction performance or learning rate of the learner w.r.t. the data seen so far.

What happens if the fundamental presumption of R-MAX and Bayesian RL fails that each single seen data-point will increase the agent's certainty about its model? This is the case if the transition dynamics change over time (so there is no correct stationary prior), or if we want to be able to ignore non-learnable, currently "too difficult" parts of the state space, either to the level of noise or limitations of a learning algorithm. For example, a household robot cannot learn how to repair a cupboard until it has achieved a basic understanding for the handling of tools. Such scenarios require the development of new methods and even new measures of success. Previous work into this direction emphasizes the concept of intrinsic motivation [10, 13, 12] and has shown empirical success in developmental robotics [1]. An interesting aspect about this work is its reliance on empirical measures of learning progress to drive exploration in reinforcement learning [17, 6]. However, to our knowledge this has not been made rigorous or related to fundamental methods like R-MAX or Bayesian RL.

In this paper, we aim to draw these relations and make the following contributions: (i) We propose to drive exploration in model-based RL by the estimated progress in model learning. We estimate this progress in terms of the loss over the training data used for model learning. (ii) We introduce

two algorithms based on modifications of R-MAX and the recent Bayesian exploration bonus (BEB) approach [9]. In contrast to the existing approaches, our algorithms do not have to assume correct prior knowledge or that the visitation counts translate directly to model certainty. Hence, they can also cope with changing dynamics. (iii) While our extensions are targeted at scenarios that go beyond the standard domain of stationary unstructured, finite state and action spaces, we provide a kind of theoretical sanity check of our extensions exactly under these standard assumptions: We discuss exploration guarantees under standard assumptions analogous to those of R-MAX and BEB.

In the next section, we review background work on exploration in Markov decision processes. Then we present our approaches for exploration based on an empirical estimate of the future model learning progress. Thereafter, we discuss guarantees on their exploration efficiency. Finally, we present an empirical evaluation before we conclude.

## 2    Background on Exploration

We model the interaction of an agent with its environment as a Markov decision process (**MDP**). An MDP is a discrete-time stochastic control process where at each time-step the process is in one of a fixed set $S$ of states and the agent can choose an action from a set $A$. The transition model $\mathcal{T}$ specifies the conditional transition distribution $\mathcal{T}(s' \mid a, s)$ over successor states $s'$ when executing an action $a$ in a given state $s$. In unstructured finite state and action spaces $\mathcal{T}$ can be defined by separate multinomial distributions $\mathcal{T}(s, a)$ over successor states for each state-action pair $s, a$. The agent receives rewards in states according to a function $\mathcal{R} : S \rightarrow \mathbb{R}$. A policy $\pi : S \rightarrow A$ specifies for each state the action to take. The goal of planning in an MDP is to find the optimal policy $\pi^*$ which maximizes the expected future rewards $E[R]$. The future rewards $R$ can be defined for a fixed horizon $H$, $R = \sum_{t=1}^{H} \mathcal{R}(s_t)$, or for an infinite horizon, $R = \sum_t \gamma^t \mathcal{R}(s_t)$, using a discount factor $0 < \gamma < 1$. The value of state $s$ when acting according to policy $\pi$ is defined as the expected future rewards when starting from $s$, $V^\pi(s) = E[R \mid s_1 = s, \pi]$. The optimal policy $\pi^*$ can be found by classical algorithms such as value iteration or policy iteration. In reinforcement learning, the agent does not know the transition model $\mathcal{T}$. In a model-based approach, the agent estimates $\hat{\mathcal{T}}$ from its interaction trace with the environment $\Delta = \langle s_1, r_1, a_1, \ldots, s_T, r_T \rangle$. Based on $\hat{\mathcal{T}}$ it computes (approximately) optimal policies. A simple approach to the exploitation-exploration tradeoff is $\epsilon$-greedy: the agent performs a random action for exploration with probability $\epsilon$ and exploits otherwise by executing a greedy policy with respect to $\hat{\mathcal{T}}$. If $\epsilon$ decreases over time towards 0, $\epsilon$-greedy exploration converges to $\pi^*$. However, this may take an inefficiently large number of non-optimal actions which is exponential in $|S|$ and $|A|$.

In the **PAC-MDP** (probably approximately correct) framework, the efficiency of an exploration algorithm $\mathcal{A}$ is measured by its sample complexity [7]. This is the number of time-steps when following $\mathcal{A}$ where its policy $\pi_t^\mathcal{A}$ at time $t$ is not $\epsilon$-optimal, that is, where $V^{\pi_t^\mathcal{A}}(s_t) < V^{\pi^*}(s_t) - \epsilon$. Given $\delta$ with $0 < \delta < 1$, $\mathcal{A}$ is said to be PAC-MDP efficient if with probability $1 - \delta$ its sample complexity scales polynomially in quantities describing $\mathcal{T}$ as well as in $\delta$ and $\epsilon$ (and $\gamma$). The model-based RL algorithms $E^3$ [8] and R-MAX [4] are PAC-MDP efficient exploration methods for unstructured finite state and action spaces: their sample complexity scales polynomially in $|S|$ and $|A|$. $E^3$ and R-MAX share the central concept of *known* states and actions which have been observed sufficiently often. (Counts are also used in the theoretical analysis of alternative PAC-MDP efficient algorithms like MBIE-EB [15].) If the visitation count $c(s, a)$ of a state-action pair $s, a$ is larger than some threshold $m$, the estimate $\hat{\mathcal{T}}(s, a)$ is guaranteed to be with high probability $\epsilon$-close to the true model. Following a policy in these known states achieves approximately the same rewards in the learned model $\hat{\mathcal{T}}$ and the true model $\mathcal{T}$. To drive exploration, R-MAX is "optimistic in the face of uncertainty" and assumes maximum reward $R_{max}$ in unknown states. This gives the reward function

$$\mathcal{R}^{\text{R-MAX}}(s, a) = \left\{ \begin{array}{ll} \mathcal{R}(s, a) & c(s, a) \geq m \ (s, a \text{ known}) \\ R_{max} & c(s, a) < m \ (s, a \text{ unknown}) \end{array} \right. .$$

Typically, the threshold $m$ is very large as $E^3$ and R-MAX need to account for all possible model instantiations within the model class of $\mathcal{T}$ [14]. For instance, for conservative choices $\epsilon = 0.1$, $\gamma = 0.9$, $\delta = 0.1$, $S = 10$, $A = 5$, we get $m > 10^6$, which is unfeasible in practice. PAC-MDP approaches like R-MAX ignore the current empirical progress in learning $\mathcal{T}$: the threshold $m$ is fixed

a-priori and remains the same for all $s, a$ independently of the agent's experiences or its estimated relevance of $s, a$ for large rewards.

**Bayesian reinforcement learning** [11] is an alternative approach to exploration. Here, the agent takes its uncertainty about the learned model $\hat{\mathcal{T}}$ explicitly into account. This allows to incorporate prior knowledge. More formally, the agent maintains a posterior belief $b$ over all possible transition models $\mathcal{T}$ given its previous experiences $\Delta$ and a prior. The value function for a deterministic policy $\pi(b, s)$ is defined in terms of the state $s$ and the belief state $b$ and fulfills

$$V^\pi(b, s) = \mathcal{R}(s, \pi(b, s)) + \int_{b', s'} \mathcal{T}(b', s' \mid b, s, \pi(b, s))\, V^\pi(b', s')\,. \qquad (1)$$

The optimal Bayesian policy $\pi^* = \mathrm{argmax}_\pi V^\pi(b, s)$ solves the exploitation-exploration tradeoff implicitly: $\pi^*$ considers how actions affect not only the state of the world, but also the agent's internal belief about the world. In a Bayesian RL context for a finite horizon $H$, the sample complexity of an algorithm $\mathcal{A}$ can be defined as the number of time-steps when following $\mathcal{A}$ where its policy $\pi_t^\mathcal{A}$ at time $t$ is not $\epsilon$-Bayesian-optimal, that is, where $V_H^{\pi_t^\mathcal{A}}(b_t, s_t) < V_H^{\pi^*}(b_t, s_t) - \epsilon$. Due to the complexity of the belief state, however, Bayesian RL is typically intractable in terms of both planning and updating the belief after an action. A recent approximate solution to Bayesian RL for unstructured finite representations is the Bayesian exploration bonus (BEB) [9] which resembles closely MBIE-EB [15]. In this setting, the belief $b$ over $\mathcal{T}$ can be expressed by means of a separate Dirichlet distribution $\alpha(s, a, s')$ for each $s, a$ with a mean estimator $\hat{\mathcal{T}}_b(s' \mid s, a) = \frac{\alpha(s, a, s')}{\sum_{s'} \alpha(s, a, s')}$.

BEB avoids reasoning in the belief space: it solves an MDP built from the mean estimate $\hat{\mathcal{T}}_b$ using an additional exploration bonus $\beta/(1 + c(s, a))$ to reward state-action pairs inversely according to their visitation counts $c(s, a)$. In the undiscounted, fixed horizon $H$ formulation BEB has a Bayesian sample complexity which with probability $1 - \delta$ is polynomial in $|S|$ and $|A|$ when setting $\beta = 2H^2$ and updating the belief for $s, a$ is stopped once $\sum_{s'} \alpha(s, a, s') > 4H^3/\epsilon$ [9].

Bayesian RL approaches are vulnerable to incorrect priors. Fard and Pineau [5] have combined Bayesian and PAC approaches to derive bounds on the approximation error in the value function of policies regardless of the correctness of the prior. However, their bounds do not apply to changing transition dynamics and it remains unanswered how to incorporate them in efficient exploration algorithms.

In a wider context of RL and developmental robotics, many strategies for efficient exploration have been subsumed by the concept of **intrinsic motivation** [10, 13] which is also termed fun or curiosity [12]. Many of these approaches take empirical learning progress into account. This includes methods that estimate from the agent's experience the amount of potential learning progress in different regions of the state space. Thereby, exploration focuses on those areas where learning progress can indeed be made: areas which are neither already well-understood nor currently too difficult to learn. The resulting algorithms enable an agent to develop progressively more complex skills. For instance, this has been demonstrated for learning in robot control [1] and options learning in hierarchical reinforcement learning [16] and has also been considered lately in machine learning under the name of curriculum learning [3]. Under the RL formalism only very recently have the concept of intrinsic motivation been mixed with standard exploration-exploitation strategies [6]. So far, however, guarantees about the sample complexity of intrinsic motivation based exploration approaches have been missing. In this paper, we take up the ideas of intrinsic motivation to extend the theoretically founded exploration approaches described above.

## 3 Exploration by Empirically Estimated Model Accuracy and Learning Progress

PAC-MDP approaches like R-MAX and Bayesian RL approaches like BEB have been developed in the context of stationary finite state-action domains. In those problems, we know that after a fixed number of visits to a state its estimated transition model becomes approximately correct and we can perform exact belief updates to guarantee Bayesian optimality. In the following, we present extensions which rely instead on previous exploration and learning experience to estimate in which parts of the state and action space further exploration is promising and where not. This is helpful in situations where the basic assumptions of R-MAX and BEB about model improvement might be violated: for example, when we have an incorrect prior (large misspecified priors may impair the

performance of BEB[9]), when we cannot come up with an a-priori threshold on the data number for learning accurate models (e.g. in domains where a polynomial KWIK learner is not available), or when the transition dynamics change over time. We can also see our approach as a method to adjust the standard R-MAX threshold online, allowing to adapt to different levels of noise on different parts of the state space.

## 3.1 Exploration Driven by Learning Progress

Let $\zeta : S \times A \mapsto \mathbb{R}$ denote a measure for the expected learning and exploration progress when visiting a state-action pair $s, a$. We discuss concrete definitions of $\zeta$ later. Clearly, $\zeta$ is a non-stationary function which changes with new experiences. Hence, an exploration strategy based on $\zeta$ needs to re-estimate $\zeta$ with each new experience. We use $\zeta$ to define two exploration algorithms.

Our first approach **$\zeta$-R-MAX** is based on R-MAX [4]. $\zeta$-R-MAX acts greedily with respect to the optimal policy for the reward function

$$\mathcal{R}^{\zeta\text{-R-MAX}}(s, a) = \begin{cases} \mathcal{R}(s, a) & \zeta(s, a) < m \\ R_{max} & \text{else} \end{cases} . \tag{2}$$

Instead of rewarding arbitrary states with low visitation counts (considered *unknown*) directly as in R-MAX, $\zeta$-R-MAX gets large reward for exploring such state-action pairs where the expected learning progress is large.

Our second approach $\zeta$-EB is based on Bayesian Exploration Bonus (BEB) [9]. $\zeta$-EB acts greedily with respect to the optimal policy for the reward function

$$\mathcal{R}^{\zeta\text{-EB}}(s, a) = \mathcal{R}(s, a) + \frac{\beta}{1 + \frac{1}{\sqrt{\zeta(s,a)}}} \tag{3}$$

for some constant $\beta$. Instead of setting the exploration bonus directly proportional to visitation counts as in BEB , $\zeta$-EB gets a bonus for exploring state-actions pairs where the expected learning progress is large. The idea of using expected learning progress to drive exploration is that we can estimate $\zeta$ empirically from the interaction data $\Delta = \langle s_1, a_1, r_1, s_2, \ldots, s_T, r_T \rangle$.

## 3.2 Empirically Estimated Model Accuracy and Learning Progress

We start by considering an empirical estimate of the current model accuracy. In a classical learning context, model accuracy is ideally tested on a held-out test data set. However, to exploit the full available data for model selection and algorithm comparison, cross validation methods have become a standard method. The work [2] discusses the importance of estimating the variance of a cross-validation estimator of the model accuracy, for instance to include this uncertainty of model accuracy in a model selection criterion. We base our following treatment of $\zeta$ on this previous work.

Let $\hat{\mathcal{T}}$ denote the estimated transition model based on data $\Delta$, which approximates the true model $\mathcal{T}$. In general, we assume that learning $\hat{\mathcal{T}}(s, a)$ implies minimizing the loss $\mathcal{L}(\hat{\mathcal{T}}(s, a); D_{s,a})$ where $D_{s,a} = \{s'_i\}_{i=1}^{n_{s,a}}$ are the successor states in the transitions $(s, a, s'_i)$ from $s, a$ in $\Delta$. $\hat{\mathcal{T}}$ may generalize over states and actions, in which case the data for learning $\hat{\mathcal{T}}(s, a)$ and evaluating the loss may include experience sets $D_{s',a'}$ with $s' \neq s, a' \neq a$ (this is for example important for relational or continuous domains). A typical loss is the neg-log data-likelihood,

$$\mathcal{L}(\hat{\mathcal{T}}; D_{s,a}) = -\frac{1}{|D_{s,a}|} \log \prod_{s' \in D_{s,a}} \hat{\mathcal{T}}(s' \mid s, a) . \tag{4}$$

Given such a loss, the predictive error is defined as

$$PE(s, a) = E_{s' \sim \mathcal{T}(s,a)} \mathcal{L}(\hat{\mathcal{T}}; \{s'\}) . \tag{5}$$

An empirical estimator of the PE based on the available data $D_{s,a}$ is the leave-one-out cross-validation estimator

$$CV(D_{s,a}, s, a) = \frac{1}{|D_{s,a}|} \sum_{s' \in D} \ell_{s'}^{loo} , \quad \ell_{s'}^{loo} := \mathcal{L}(\hat{\mathcal{T}}^{-s'}; \{s'\}) \tag{6}$$

where $\hat{\mathcal{T}}^{-s'}$ is the model learned from data $D_{s,a}^{-s'} = D_{s,a} \setminus \{s'\}$.

Putting an absolute threshold directly on the loss to decide whether a state is known or unknown is hard. Note that the predictive error $PE(s,a) = \mathrm{KL}\big(\mathcal{T}(s,a) \,\big\|\, \hat{\mathcal{T}}(s,a)\big) + \mathrm{H}\big(\mathcal{T}(s,a)\big)$ has the entropy of the true distribution as a lower bound, which is unknown. Therefore, we propose to drive exploration based on the learning *progress* instead of the current learner accuracy. Using the change in loss we may gain robustness by becoming independent of the loss' absolute value and can potentially detect time-varying conditions.

We define $\zeta$ in terms of the change in the (empirically estimated) loss as follows. Let $D_{s,a}^{-k} = \{s_i' \in D_{s,a} \mid i < n_{s,a} - k\}$ denote the experiences in $D_{s,a}$ except the last $k$. $\hat{\mathcal{T}}^{-k}$ is the transition model learned from the reduced data-set $D_{s,a}^{-k}$, in contrast to $\hat{\mathcal{T}}$ which is learned from all data $D_{s,a}$. We define

$$\hat{\zeta}(s,a) := CV(D_{s,a}^{-k}, s, a) - CV(D_{s,a}, s, a) \approx \mathcal{L}(\hat{\mathcal{T}}^{-k}; D_{s,a}) - \mathcal{L}(\hat{\mathcal{T}}; D_{s,a}) \,. \tag{7}$$

This estimates to which extent the last $k$ experiences help to learn a better model as evaluated over the complete data. Thus, if $\hat{\zeta}(s,a)$ is small, then the last $k$ visitations in the data-set $D_{s,a}$ did not have a significant effect on improving $\hat{\mathcal{T}}$ and in turn $s, a$ does not require further exploration.

The estimator $\hat{\zeta}(s,a)$ defined above is only a mean estimator of the learning progress. The $\zeta(s,a)$ we use in concrete exploration algorithms $\zeta$-EB and $\zeta$-R-MAX includes an additional variance margin,

$$\zeta(s,a) := \hat{\zeta}(s,a) + \alpha \sqrt{\nu(s,a)} \,, \tag{8}$$

where $\nu(s,a)$ is an estimate of the CV variance (discussed in more detail below),

$$\nu(s,a) = \frac{1}{|D_{s,a}|} \sum_{s' \in D_{s,a}} [\ell_{s'}^{loo} - CV(D_{s,a}, s, a)]^2 \,. \tag{9}$$

The variance margin increases robustness and is motivated by the following analysis.

## 3.3 Guarantees on the Exploration Efficiency

As discussed in the introduction, we propose our extensions of R-MAX and BEB to target scenarios which go beyond the standard setting of stationary unstructured, finite state and action spaces. In this subsection, however, we go back and consider the behavior of our extensions exactly under these classical assumptions—this is meant as a sanity check to ensure that our extensions inherit the standard exploration efficiency properties under standard assumptions. We will directly relate a threshold on the empirical $\zeta(s,a)$ to a threshold on model accuracy.

We start by providing two properties of the *mean* $\left\langle \hat{\zeta}(s,a) \right\rangle_{D_{s,a}}$ under random data. First we find that the expected $\hat{\zeta}(s,a)$ converges with $1/n^2$:

**Lemma 1.** *For a Dirichlet learner in a stationary environment, we have*

$$\left\langle \hat{\zeta}(s,a) \right\rangle_{D_{s,a}} = O\left(\frac{1}{n_{s,a}^2}\right) \,.$$

A proof is given in the supplementary material. Similarly, a threshold on the mean $\left\langle \hat{\zeta}(s,a) \right\rangle_{D_{s,a}}$ implies a model accuracy threshold:

**Lemma 2.** *Given an approximated model $\hat{\mathcal{T}}$ of a true model $\mathcal{T}$, for any $\epsilon$ there exists an $\epsilon'$ such that:*

$$\left| \left\langle \hat{\zeta}(s,a) \right\rangle_{D_{s,a}} \right| < \epsilon' \quad \Rightarrow \quad |\hat{\mathcal{T}}(s,a) - \mathcal{T}(s,a)| < \epsilon \,. \tag{10}$$

*Sketch of proof.* For the case of multinomial variables, we know that the maximum likelihood estimator is consistent and unbiased and is equal to the normalized visit counts. In this situation we know that as $n \to \infty$, $\sqrt{n}(\hat{p} - p) \rightsquigarrow N(0, \Sigma)$. Our measure of progress $\zeta$ is the difference between

two maximum likelihood estimators and so, as $n$ approaches infinity we have the same limiting result on the model quality, with a higher variance due to the subtraction between two different random variables and the correlation between them. □

With the previous results we know that if we had access to the mean $\left\langle \hat{\zeta}(s,a) \right\rangle_{D_{s,a}}$ under random data we would be able to assess model error by looking at its value. Unfortunately, the agent only has access to an empirically estimate. To ensure that we can define robust criteria for considering a state as known, we have to consider the variability of the estimator $\hat{\zeta}(s,a)$ under random data $D_{s,a}$. As discussed in [2], the estimator is unbiased, that is $\langle CV(D_{s,a},s,a) \rangle_D = PE(s,a)$ and, in the limit $|D_{s,a}| \to \infty$, the $(\hat{\mathcal{T}} - \mathcal{T})$ becomes Gaussian. Its variance can be described by first considering the covariance matrix $\mathcal{C}$ of the vector $(\ell_{s'}^{loo})_{s' \in D_{s,a}}$ under random $D_{s,a}$. The diagonal entries are the variances $\mathrm{Var}_D\{\ell_{s'}^{loo}\}$ of each single $\ell_{s'}^{loo}$ under random data, which are independent of $s'$ (assuming i.i.d. data) and therefore equally estimated as $\nu(s,a)$ given in Eq. (9). The off-diagonal entries of $\mathcal{C}$ capture the correlations between different $\ell_{s'}^{loo}$ and $\ell_{s''}^{loo}$ and are constant (due to i.i.d. data; see [2] for details). By assuming these correlations to vanish we over-estimate the CV variance and therefore have, from Eq. (6),

$$\mathrm{Var}_{D_{s,a}}\{CV(D_{s,a},s,a)\} \le \nu(s,a) \tag{11}$$

Having an overestimation of the variance of the loss we will consider what is the variance of the estimation of $\hat{\zeta}(s,a)$. Both terms $\mathcal{L}(\hat{\mathcal{T}};D_{s,a})$ and $\mathcal{L}(\hat{\mathcal{T}}^{-k};D_{s,a})$ are estimated using LOO-CV, allowing us to bound the $\hat{\zeta}$'s variance under random data, from Eq. (11), as:

$$\mathrm{Var}_{D_{s,a}}\{\hat{\zeta}(s,a)\} \le 2\nu(D_{s,a},s,a) . \tag{12}$$

Now that we have a confidence measure on the estimator we can show that a threshold on the empirical estimator $\zeta(s,a) = \hat{\zeta}(s,a) + \alpha\sqrt{\nu(s,a)}$ implies a threshold on the mean $\left\langle \hat{\zeta}(s,a) \right\rangle_{D_{s,a}}$:

**Lemma 3.** *For any given $\delta$ with $0 < \delta < 1$ and $\epsilon > 0$ there exists an $\alpha$ such that*

$$\left[|\hat{\zeta}(s,a)| + \alpha \sqrt{\nu(s,a)} < \epsilon \Rightarrow |\left\langle \hat{\zeta}(s,a) \right\rangle_{D_{s,a}}| < \epsilon\right] \text{ with probability } 1 - \delta . \tag{13}$$

*Proof.* For a Gaussian variable $x$ with mean $\mu$ and variance $\nu$ we know that $x < \mu + \alpha\sqrt{\nu}$ with probability given by the error function $\delta = 1/2 + \mathrm{erf}(\alpha/\sqrt{2})/2$. By inverting this we get $\alpha$ to fulfill the above. □

Finally, we show that our exploration method using the empirical $\zeta(s,a)$ to drive exploration is PAC-MDP efficient.

**Lemma 4.** *There is a threshold $m$ such that $\zeta$-R-MAX using a Dirichlet learner in the standard setting of stationary unstructured, finite state and action spaces is PAC-MDP efficient.*

*Proof.* From Lemma 3 we know that a threshold on our empirical measure implies a threshold on the mean measure. From Lemma 2 we know that a small $\left\langle \hat{\zeta}(s,a) \right\rangle_{D_{s,a}}$ corresponds, with high probability, to a low model error. Under these conditions a state is only marked as known if the empirical measure is below a certain threshold; this ensures with high probability that the error in the model for the state-action is low. From the standard conditions of R-MAX, $\zeta$-R-MAX is PAC-MDP efficient. □

## 4  Evaluation

We compare the empirical performance of our exploration algorithms $\zeta$-R-MAX and $\zeta$-EB with the performance of R-MAX, BEB and simple model-based $\epsilon$-greedy exploration with optimistic initialization in unstructured finite state and action spaces. We investigate different scenarios where the assumptions of R-MAX and BEB are fulfilled or violated. We define these scenarios by varying the level of stochasticity in state transitions. BEB assumes to have a correct a prior about this stochasticity. R-MAX assumes to know correct thresholds $m$ for the number of visits to states to

ensure accurate transition models. We simulate satisfied or violated assumptions of R-MAX by setting individual thresholds for states: in general, states with higher noise require more samples to achieve the same level of model accuracy as states with low noise. Setting individual thresholds is equivalent to setting individual initials counts (instead of 0s) for states.

We investigate three questions: (a) How close do our algorithms $\zeta$-R-MAX and $\zeta$-EB get to the performance of the original algorithms when the assumptions of the latter are correct? (b) How much do $\zeta$-R-MAX and $\zeta$-EB gain if the assumptions of the original algorithms are violated? (c) And are our approaches more robust to unexpected changes in the transition dynamics?

Our evaluation environment (shown in Fig. 1(a)) is a discrete MDP with **25** states and five actions: **up**, **down**, **left**, **right** and **stop**. There is a single goal state with a reward of 1 (marked with "G") and several states with negative rewards $-0.1$ (darker states). The lighter states are noisy states: their actions have less predictable effects. The transition probabilities of the noisy states are sampled from a Dirichlet distribution with parameters $\alpha = 0.1$, while the probabilities for all other states are sampled from a Dirichlet with $\alpha = 1.0$, in both cases eventually permuted to ensure that the highest probability corresponds to the expected next state according to the name definition of the actions. The shortest path from start to goal is not optimal due to the uncertainty in the transitions. Instead, the optimal path avoids the lighter states and takes the route below. To find this optimal path and avoid local minima, an exploration algorithm needs to explore sufficiently and estimate the state values accurately. We evaluate the performance of the algorithms in terms of the reward collected in the true model $\mathcal{T}$ using the optimal policy $\pi^*_{\hat{\mathcal{T}}}$ for their learned model $\hat{\mathcal{T}}$. In our results, we report the policy value error defined as $V_{\mathcal{T}}(s_I; \pi^*_{\hat{\mathcal{T}}}) - V_{\mathcal{T}}(s_I; \pi^*_{\mathcal{T}})$ in the value of the start state $s_I$ with respect to the optimal policy $\pi^*$. In our experiments, the agent is reset to $s_I$ every 30 steps. All figures presented in the following show the means and standard deviations over 20 runs. For the $\zeta$ estimation we use $k = 10$.

**Experiment 1: Correct Assumptions**  In our first experiment, the assumptions of BEB and R-MAX are fulfilled: BEB is given the correct prior; R-MAX uses appropriate thresholds for states (depending on the state stochasticity). $\zeta$-R-MAX, $\zeta$-EB and $\epsilon$-greedy are not given any knowledge. The results presented in Fig. 1(b) show that our exploration methods $\zeta$-R-MAX and $\zeta$-EB achieve similar performance to the original methods, even without having a correct prior or state-dependent thresholds. Both $\zeta$-R-MAX and $\zeta$-EB converge to the correct final policy, requiring only moderately more steps. In contrast, $\epsilon$-greedy does not find the optimal policy in reasonable time. Clearly, the original algorithms could also be executed based on likewise correct, but less informative assumptions: BEB with an uninformative prior, R-MAX with more conservative (larger) uniform threeholds. Then, R-MAX would need longer learning time; BEB might not converge, see [9].

**Experiment 2: Violated Assumptions**  Here, the assumptions of R-MAX and BEB are violated (wrong thresholds/priors). This may well happen in practice where correct priors cannot always be specified or the counts cannot be translated directly to model certainty. In each run, we initialize R-MAX and BEB with a random uniform prior (in the interval of the minimum and maximum values of the true prior, translated to counts for R-MAX). The results in Fig. 1(c) show that as expected R-MAX and BEB do not converge any longer to the optimal policy: they explore states too long whose dynamics are already well estimated, while neglecting states which require more samples for an accurate model estimation. In contrast, $\zeta$-R-MAX and $\zeta$-EB do not rely on these assumptions and again converge to the optimal policy.

**Experiment 3: Change in Dynamics**  In our final experiment, the transition dynamics for a randomly chosen state along the optimal path get permuted after 900 steps. As Fig. 1(d) shows, R-MAX and BEB with correct assumptions for the original problem (before time-step 900) cannot compensate for this as they base their estimate of the model certainty only on the visitation counts, but do not look at the data itself. In contrast, $\zeta$-R-MAX and $\zeta$-EB detect such an unexpected event and can refocus their exploration efforts.

## 5   Conclusions and Future Work

We have proposed to drive exploration in model-based reinforcement learning using the estimated future progress in model learning. When estimating this learning progress empirically, exploration

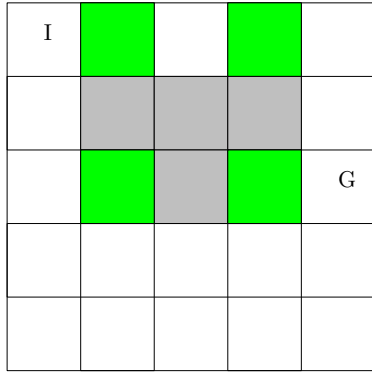

(a) *Evaluation Environment*

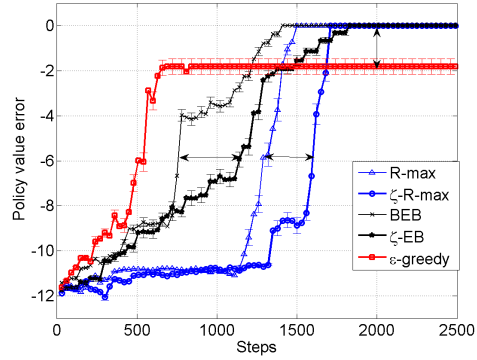

(b) *Experiment 1—Correct Assumptions*

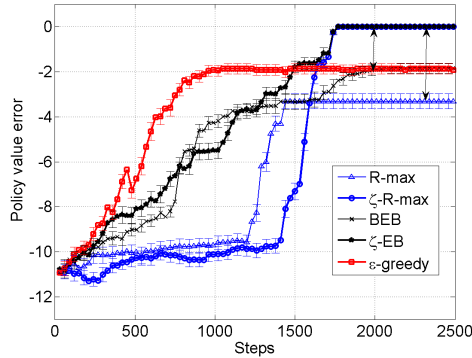

(c) *Experiment 2—Violated Assumptions*

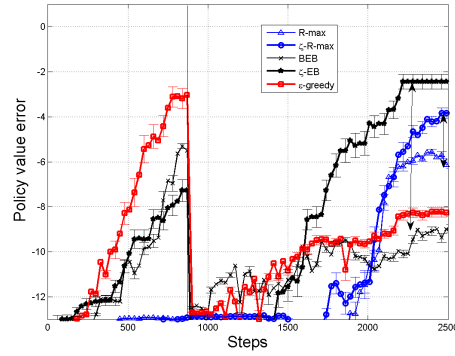

(d) *Experiment 3—Change in Dynamics*

Figure 1: *Experiments*: *(a)* The agent starts at state "I" and needs to get to goal "G". Grey (darker) states incur a negative reward. Green (lighter) states have very noisy transition dynamics. *(b)* Like R-MAX and BEB with correct assumptions, our algorithms $\zeta$-R-MAX and $\zeta$-EB based on an empirical estimation of the learning progress converge to the optimal policy without relying on these assumptions, but take a small extra amount of time. *(c)* When their assumptions are violated, R-MAX and BEB fail to converge, while $\zeta$-R-MAX and $\zeta$-EB don't rely on these assumptions and again find the optimal policy. *(d)* In contrast to existing methods, $\zeta$-R-MAX and $\zeta$-EB can cope with the change in transition dynamics after 900 steps and refocus their exploration.

algorithms can be defined which do not rely on correct prior knowledge and can cope with changing transition dynamics. As a theoretical "sanity check" we have discussed efficiency guarantees of our approaches similar to the ones of the established algorithms. Our novel problem settings provide interesting opportunities for the development of RL algorithms and theoretical analyses for relevant real-world scenarios, in particular for structured, continuous and non-stationary domains. It is also worth to investigate in more depth the relation of our approach to the general concept of intrinsic motivation as proposed in developmental robotics. In our view, a combination of methods which trades off both strong prior assumptions together with empirical estimates of the learning progress seems to be the most promising direction for future work on exploration in the real world.

## Acknowledgments

Work supported by the Flowers Team (INRIA/ENSTA-Paristech), Conseil Régional d'Aquitaine and the ERC grant EXPLORERS 24007. Tobias Lang and Marc Toussaint were supported by the German Research Foundation under grants TO 409/1-3 and TO 409/7-1.

# References

[1] A. Baranes and P.Y. Oudeyer. Active learning of inverse models with intrinsically motivated goal exploration in robots. *Robotics and Autonomous Systems*, 2012.

[2] Yoshua Bengio and Yves Grandvalet. No unbiased estimator of the variance of k-fold cross-validation. *Journal of Machine Learning Research (JMLR)*, 5:1089–1105, 2004.

[3] Yoshua Bengio, Jérôme Louradour, Ronan Collobert, and Jason Weston. Curriculum learning. In *Int. Conf. on Machine Learning (ICML)*, pages 41–48, 2009.

[4] Ronen I. Brafman and Moshe Tennenholtz. R-max - a general polynomial time algorithm for near-optimal reinforcement learning. *Journal of Machine Learning Research (JMLR)*, 3:213–231, 2002.

[5] Mahdi Milani Fard and Joelle Pineau. Pac-bayesian model selection for reinforcement learning. In *Conf. on Neural Information Processing Systems (NIPS)*. 2010.

[6] Todd Hester and Peter Stone. Intrinsically motivated model learning for a developing curious agent. In *AAMAS Workshop on Adaptive Learning Agents (ALA)*, 2012.

[7] Sham Kakade. *On the Sample Complexity of Reinforcement Learning*. PhD thesis, Gatsby Computational Neuroscience Unit, University College London, 2003.

[8] Michael Kearns and Satinder Singh. Near-optimal reinforcement learning in polynomial time. *Machine Learning Journal*, 49(2-3):209–232, 2002.

[9] J. Zico Kolter and Andrew Ng. Near-Bayesian exploration in polynomial time. In *Int. Conf. on Machine Learning (ICML)*, pages 513–520, 2009.

[10] P.Y. Oudeyer, F. Kaplan, and V.V. Hafner. Intrinsic motivation systems for autonomous mental development. *IEEE Transactions on Evolutionary Computation*, 11(2):265–286, 2007.

[11] Pascal Poupart, Nikos Vlassis, Jesse Hoey, and Kevin Regan. An analytic solution to discrete Bayesian reinforcement learning. In *Int. Conf. on Machine Learning (ICML)*, 2006.

[12] Jürgen Schmidhuber. Curious model-building control systems. In *Proc. of Int. Joint Conf. on Neural Networks*, volume 2, pages 1458–1463, 1991.

[13] Satinder Singh, Andrew G. Barto, and Nuttapong Chentanez. Intrinsically motivated reinforcement learning. In *Conf. on Neural Information Processing Systems (NIPS)*, pages 1281–1288. 2005.

[14] Alexander L. Strehl, Lihong Li, and Michael Littman. Reinforcement learning in finite MDPs: PAC analysis. *Journal of Machine Learning Research (JMLR)*, 2009.

[15] Alexander L. Strehl and Michael L. Littman. An analysis of model-based interval estimation for markov decision processes. *J. Comput. Syst. Sci.*, 74(8):1309–1331, 2008.

[16] Christopher M. Vigorito and Andrew G. Barto. Intrinsically motivated hierarchical skill learning in structured environments. *IEEE Transactions on Autonomous Mental Development (TAMD)*, 2(2), 2010.

[17] Marco Wiering and Jürgen Schmidhuber. Efficient model-based exploration. In *International Conference on Simulation of Adaptive Behavior: From Animals to Animats 6*, 1998.

